# Multi-electrode spike sorting by clustering transfer functions

Dmitry Rinberg        Hanan Davidowitz        Naftali Tishby*

NEC Research Institute
4 Independence Way
Princeton, NJ 08540
E-mail: {dima,hanan,tishby}@research.nj.nec.com
Categories: spike sorting, population coding, signal processing.

## Abstract

A new paradigm is proposed for sorting spikes in multi-electrode data using ratios of transfer functions between cells and electrodes. It is assumed that for every cell and electrode there is a stable linear relation. These are dictated by the properties of the tissue, the electrodes and their relative geometries. The main advantage of the method is that it is insensitive to variations in the shape and amplitude of a spike. Spike sorting is carried out in two separate steps. First, templates describing the statistics of each spike type are generated by clustering transfer function ratios then spikes are detected in the data using the spike statistics. These techniques were applied to data generated in the escape response system of the cockroach.

## 1 Introduction

Simultaneous recording of activity from many neurons can greatly expand our understanding of how information is coded in neural systems[1]. Multiple electrodes are often used to measure the activity in neural tissue and have become a standard tool in neurophysiology [2, 3, 4]. Since every electrode is in a different position it will measure a different contribution from each of the different neurons. Simply stated, the problem is this: how can these complex signals be untangled to determine when each individual cell fired? This problem is difficult because, a) the objects being classified are very similar and often noisy, b) spikes coming from the same cell can

vary in both shape and amplitude, depending on the previous activity of the cell and c) spikes can overlap in time, resulting in even more complex temporal patterns.

Current approaches to spike sorting are based primarily on the presumed consistency of the spike shape and amplitude for a given cell [5, 6]. This is clearly the only possible basis for sorting using a single electrode. Multiple electrodes, however, provide additional *independent* information through the differences in the way the same neuron is detected by the different electrodes. The same spike measured on different electrodes can differ in amplitude, shape and its relative timing. These differences can depend on the specific cell, the electrode and the media between them. They can be characterized by linear transfer functions that are invariant to changes in the overall spike waveform. In this paper the importance of this information is highlighted by using *only* the differences in how signals are measured on different electrodes. It is then shown that clusters of similar *differences* correspond to the same neuron. It should be emphasized that in a full treatment this transfer function information will be combined with other cues to sort spikes.

## 2  Spikes, spectra and noise

The basic assumption behind the spike sorting approach described here is that the medium between each neuron-electrode pair can be characterized by a linear system that remains fixed during the course of an experiment. This assumption is justified by the approximately linear dielectric properties of the electrode and its surrounding nerve tissues.

Linear systems are described by their phase and amplitude response to pure frequencies, namely, by their complex transfer function $H(\omega) = O(\omega)/I(\omega)$, where $I(\omega)$ and $O(\omega)$ are the complex spectra (i.e. Fourier transform, henceforth called spectrum) of the input and output of the system, respectively. In the experiments described here the input signal is the spectrum of the action potential generated by cell $j$, denoted by $S_j(\omega)$ and the output signal is the spectrum of the voltage measured at electrode $\mu$, denoted by $V^\mu(\omega)$. The transfer function of the system that links $S_j(\omega)$ and $V^\mu(\omega)$ is then defined as $H_j^\mu(\omega) = V^\mu(\omega)/S_j(\omega)$.

If the transfer functions are fixed in time, the ratio between the complex spectrum of any spike from cell $j$ as detected by electrodes $\mu$ and $\nu$, $V^\mu(\omega)$ and $V^\nu(\omega)$, is given by,

$$T_j^{\mu\nu}(\omega) \equiv \frac{V^\mu(\omega)}{V^\nu(\omega)} = \frac{H_j^\mu(\omega)S_j(\omega)}{H_j^\nu(\omega)S_j(\omega)} = \frac{H_j^\mu(\omega)}{H_j^\nu(\omega)} \, , \qquad (1)$$

which is independent of the cell action potential spectrum $S_j(\omega)$, provided that the spike was detected by both electrodes.

Thus, even if a spike varies in shape and amplitude, $T_j^{\mu\nu}(\omega)$ will remain a fixed complex function of frequency. This ratio is also invariant with respect to time translations of the spikes. In addition, the frequency components are asymptotically uncorrelated for stationary processes, which justifies treating the frequency components as statistically independent[7]. The idea behind the approach described here is shown in Figure 1.

In real experiments, however, noise can corrupt the invariance of $T_j^{\mu\nu}$. There are several possible sources of noise in experiments of this kind: a) fluctuations in the transfer function, b) changes in the spike shape, $\varsigma_j$ and c) electrical and electrochemical noise, $n^\mu$.

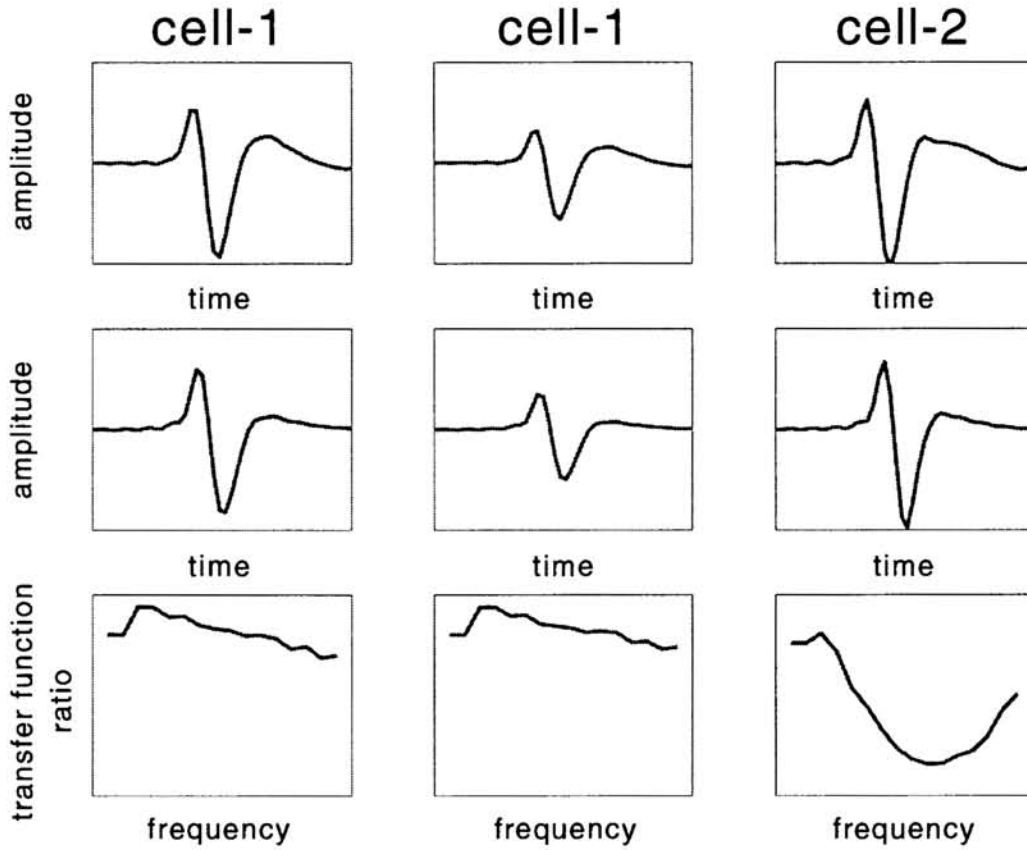

Figure 1: The idea behind spike sorting by clustering of transfer function ratios. Two spikes from the same cell (cell-1) may vary in shape/amplitude during bursting activity, for example. Although the spike shapes may differ, the transfer functions relating them to the electrodes do not change so the transfer function ratios are similar (two left columns). A different cell (cell-2) has a different transfer function ratio even though the spikes shapes themselves are similar to those of cell-1 (right column).

If $H_j^\mu$ varies slowly with time, the transfer function noise is small relative to $\varsigma_j$, $n^\nu$ and $n^\nu$. $T_j^{\mu\nu}$ can then be expanded to first order in $\varsigma_j$, $n^\mu$ and $n^\nu$ as

$$T_j^{\mu\nu}(\omega, t) = \frac{H_j^\mu(S_j + \varsigma_j) + n^\mu}{H_j^\nu(S_j + \varsigma_j) + n^\nu} = \frac{H_j^\mu}{H_j^\nu}\left(1 + \frac{n^\mu}{H_j^\mu S_j} - \frac{n^\nu}{H_j^\nu S_j}\right), \qquad (2)$$

which is independent of $\varsigma_j$. Since the noise, $n^\mu$, is uncorrelated with the spike signal, $S_j$, the variance at each frequency component can be considered to be Gaussian with equal variances on the real and imaginary axes. Thus the mean of $T_j^{\mu\nu}$ will be independent of $S_j, \varsigma_j$ and $n^\mu$ while its variance will be inversely proportional to $S_j$.

## 3   A model system: the escape response of the cockroach

These techniques were tested on a relatively simple neural system - the escape response system of the American cockroach. The escape behaviour, which has been studied extensively [9, 10, 11], is activated when the insect detects air currents

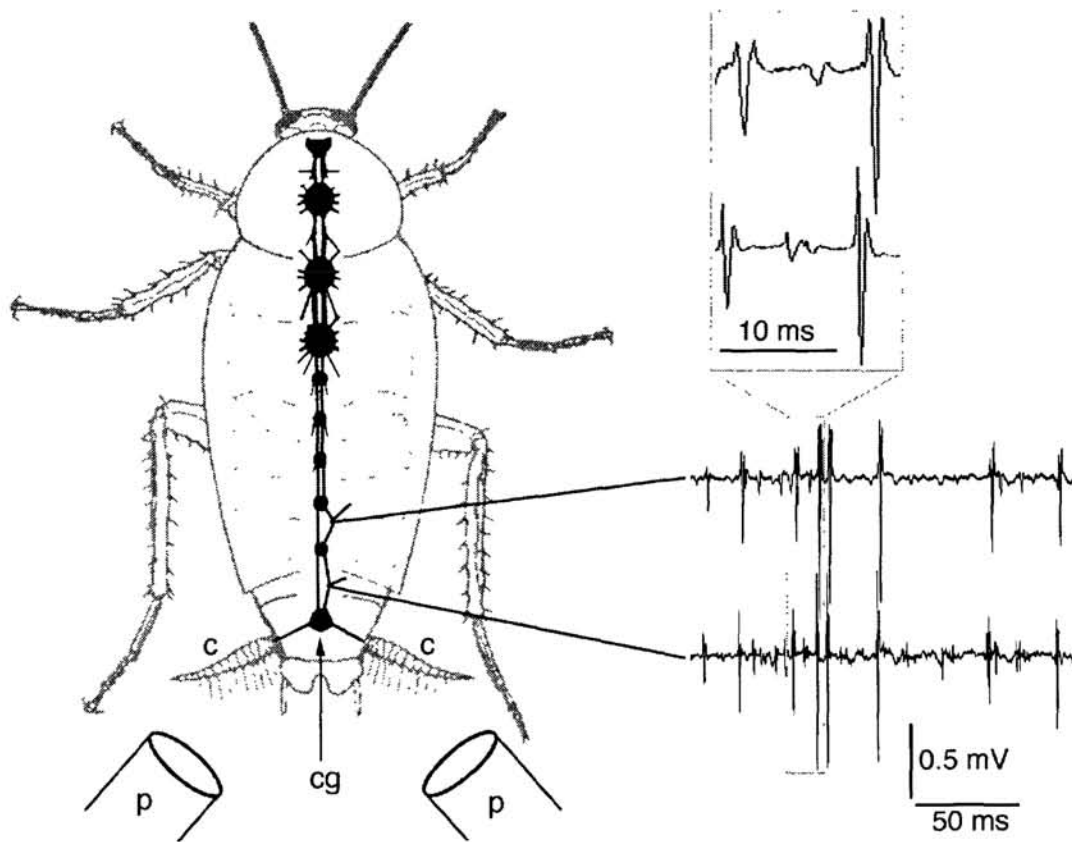

Figure 2: A schematic representation of the experiment. Typical raw data measured on two electrodes is shown at right. Relative time delays are evident in the inset, but are not a necessary condition for the sorting techniques described here. Abbreviations are: p-puffers, cg-circal ganglion, c-cerci.

produced by the movements of a predator. The insect detects the approach of a predator, determines the direction of approach and starts running in an appropriate direction. The cockroach does this by detecting the movement of several hundred fine hairs located on two appendages, called cerci, protruding from the posterior end of the animal. Each of these hairs is connected to a single neuron. Axons from these cells converge on a dense neuropil called the cercal ganglion (cg), where directional information is extracted and conveyed to the rest of the body by axons in the abdominal nerve. This is shown schematically in Figure 2.

This system proved to be well suited as a first test of the sorting technique. The system is simple enough so that it is not overwhelming (since only 7 neurons are known to contribute to the code) but complex enough to really test the approach. In addition, the nerve cords are linear in geometry, easily accessible and very stable.

Male cockroaches (*Periplaneta americana*) were dissected from the dorsal side to expose the nerve cord. The left and right cords were gently separated and two tungsten wire electrodes were hooked onto the connective about 2 mm apart, separated by abdominal ganglia. The stimulus was presented by two loudspeakers driving two miniature wind tunnels pointed at the cerci, at 90 degrees from one another as shown in Figure 2. Recordings typically lasted for several hours. Data were collected with a sampling frequency of $2 \cdot 10^4$ S/s which was sufficient to preserve the high frequency components of the spikes.

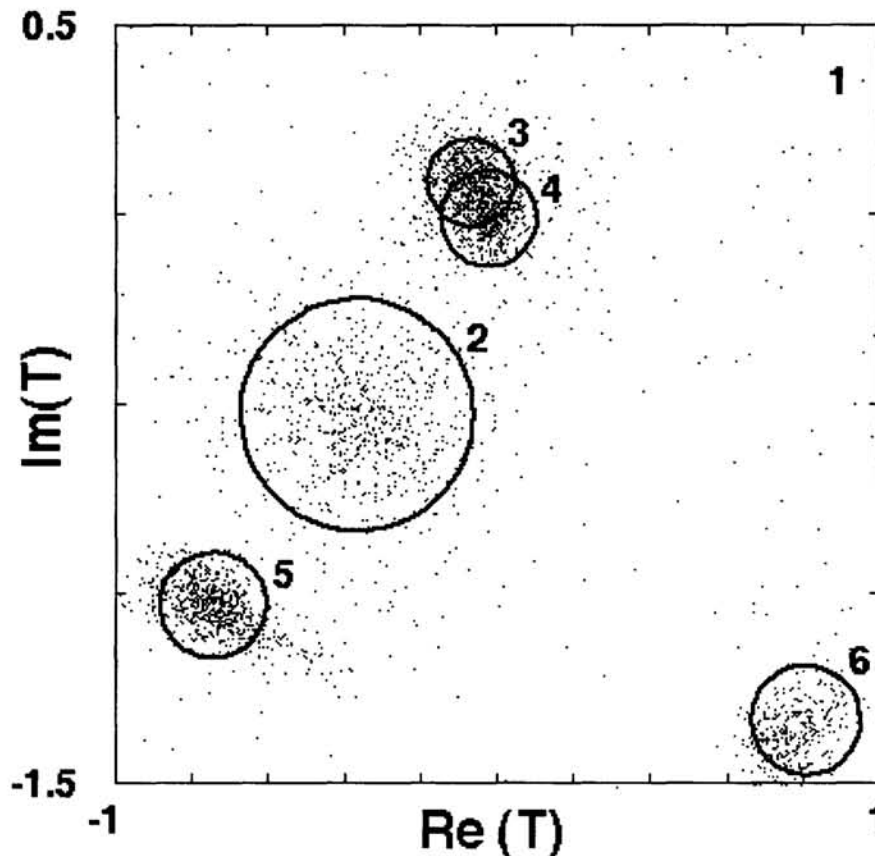

Figure 3: Real and imaginary parts of $T_j^{\mu\nu}$ a single $\omega$. The circles have centers (radii) equal to the average (variance) of $T_j^{\mu\nu}$ at $\omega = 248.7$ rad s$^{-1}$. Note that while some clusters seem to overlap at this frequency they may be well seperated at others. Cluster-1 is dispersed throughout the complex plane and its variance is well beyond the range of this plot.

## 4    Clustering and the detection of spikes

The spike sorting algorithm described here is done is two separate stages. First, a statistical model of the individual spike types is built from "clean" examples found in the data. Only then are occurrences of these spikes detected in the multi-electrode data. This two-step arrangement allows a great deal of flexibility by disconnecting the clustering from the detection. For example, here the clustering was done on transfer function ratios while the detection was done on full complex spectra. These stages are described below in more detail.

### 4.1    The clustering phase

First, the multi-electrode recording is chopped into 3 ms long frames using a sliding window. Frames that have either too low total energy or too high energy at the window edges are discarded. This leaves frames that are energetic in their central 2 ms and are assumed to carry one spike. No attempt is made to find all spikes in the data. Instead, the idea is to generate a set of candidate spike types from clean frames.

Once a large collection of candidate spikes is found, $T_j^{\mu\nu}(\omega)$ is calculated for every

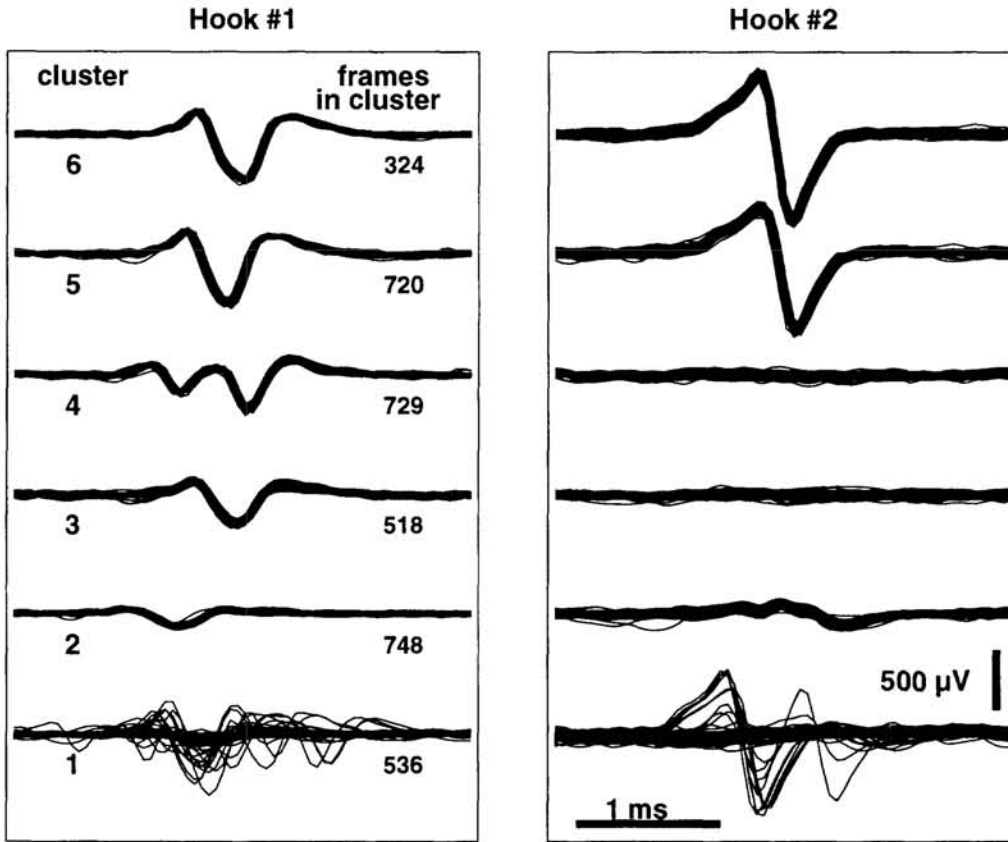

Figure 4: Results of clustering spikes using transfer function ratios. Note that although cluster-5 and cluster-6 are similarly shaped on hook-1 they are time shifted on hook-3. Cluster-1 is made up of overlaps which are dealt with in the detection phase.

spike. These are then grouped together into clusters containing similar $T_j^{\mu\nu}(\omega)$. Results of the clustering are shown in Figure 3 while the corresponding waveforms are shown in Figure 4. Full complex spectra are then used to build a statistical model of the different spike types, $\{V_j^\mu(\omega), \sigma_j^\mu(\omega)\}$, which represent each cell's action potential as it appears on each of the electrodes.

## 4.2 The detection phase

Once the cluster statistics are determined, an independent detection algorithm is used. The data is again broken into short frames but now the idea is to find which of the spike types (represented by the different clusters found in the previous steps) best represents the data in that frame. Each frame can contain either noise, a spike or an overlap of 2 spikes (overlaps of more than 2 spikes are not dealt with). This part is not done on transfer function ratios because dealing with overlaps is more difficult.

## 5  Conclusion

A new method of spike sorting using transfer function ratios has been presented. In effect the sorting is done on the properties of the tissue between the neuron and

the electrode and individual spike shapes become less important. This method may be useful when dealing with bursting cells where the transfer function ratios should remain constant even though the spike amplitude can change significantly. This technique may prove to be a useful tool for analysing multi-electrode data.

## Acknowledgments

We are grateful to Bill Bialek for numerous enlightening discussions and many useful suggestions.

## Footnotes

*Permanent address: Institute of Computer Science and Center for Neural Computation, The Hebrew University, Jerusalem, Israel. Email: tishby@cs.huji.ac.il

## References

[1] M. Barinaga. Listening in on the brain. *Science* **280**, 376-378 (1998).

[2] M. Abeles. *Corticonics*, (Cambridge University Press, Cambridge, 1991)

[3] B.L. McNaughton, J. O'Keefe and C.A. Barnes. The stereotrode: a new technique for simultaneous isolation of several single units in the central nervous system from multiple unit records. *Journal of Neuroscience Methods,* **8**, 391-7 (1983).

[4] M.L. Reece and J. O'Keefe. The tetrode: a new technique for multi-unit extracellular recording. *Society of Neuroscience Abstracts* **15**, 1250 (1989).

[5] M.S. Fee, P. P. Mitra and D. Kleinfeld. Automatic sorting of multiple unit neuronal signals in the presence of anisotropic and non-Gaussian variability. *Journal of Neuroscience Methods* **69**, 175-188 (1996).

[6] M.S. Lewicki. A review of methods for spike sorting: the detection and classification of neural potentials. *Network: Compututational Neural Systems* **9**, R53-R78 (1998).

[7] A. Papoulis. *Probability, random variables and stochastic processes*, (McGraw-Hill, New-York, 1965).

[8] M. Abeles and G.L. Gerstein. Detecting spatio-temporal firing patterns among simultaneously recorded single neurons. *Journal of Neurophysiology* **60**(3), 909-924 (1988).

[9] J.M. Camhi and A. Levy. The code for stimulus direction in a cell assembly in the cockroach. *Journal of Comparative Physiology A* **165**, 83-97 (1989).

[10] L. Kolton and J.M. Camhi. Cartesian representation of stimulus direction: parallel processing by two sets of giant interneurons in the cockroach. *Journal of Comparative Physiology A* **176**, 691-702 (1995).

[11] J. Westin, J.J. Langberg and J.M. Camhi. Responses of Giant Interneurons of the cockroach *Periplaneta americana* to wind puffs of different directions and velocities. *Journal of Comparative Physiology A* **121**, 307-324 (1977).